# Analysis of Contour Motions

**Ce Liu    William T. Freeman    Edward H. Adelson**
Computer Science and Artificial Intelligence Laboratory
Massachusetts Institute of Technology
Cambridge, MA 02139, USA
{celiu,billf,adelson}@csail.mit.edu

## Abstract

A reliable motion estimation algorithm must function under a wide range of conditions. One regime, which we consider here, is the case of moving objects with contours but no visible texture. Tracking distinctive features such as corners can disambiguate the motion of contours, but spurious features such as T-junctions can be badly misleading. It is difficult to determine the reliability of motion from local measurements, since a full rank covariance matrix can result from both real and spurious features. We propose a novel approach that avoids these points altogether, and derives global motion estimates by utilizing information from three levels of contour analysis: edgelets, boundary fragments and contours. Boundary fragment are chains of orientated edgelets, for which we derive motion estimates from local evidence. The uncertainties of the local estimates are disambiguated after the boundary fragments are properly grouped into contours. The grouping is done by constructing a graphical model and marginalizing it using importance sampling. We propose two equivalent representations in this graphical model, reversible switch variables attached to the ends of fragments and fragment chains, to capture both local and global statistics of boundaries. Our system is successfully applied to both synthetic and real video sequences containing high-contrast boundaries and textureless regions. The system produces good motion estimates along with properly grouped and completed contours.

## 1   Introduction

Humans can reliably analyze visual motion under a diverse set of conditions, including textured as well as featureless objects. Computer vision algorithms have focussed on conditions of texture, where junction or corner-like image structures are assumed to be reliable features for tracking [5, 4, 17]. But under other conditions, these features can generate spurious motions. T-junctions caused by occlusion can move in an image very differently than either of the objects involved in the occlusion event [11]. To properly analyze motions of featureless objects requires a different approach.

The spurious matching of T-junctions has been explained in [18] and [9]. We briefly restate it using the simple two bar stimulus in Figure 1 (from [18]). The gray bar is moving rightward in front of the leftward moving black bar, (a). If we analyze the motion locally, i.e. match to the next frame in a local circular window, the flow vectors of the corner and line points are as displayed in Figure 1 (b). The T-junctions located at the intersections of the two bars move downwards, but there is no such a motion by the depicted objects.

One approach to handling the spurious motions of corners or T-junctions has been to detect such junctions and remove them from the motion analysis [18, 12]. However, T-junctions are often very difficult to detect in a static image from local, bottom-up information [9]. Motion at occluding boundaries has been studied, for example in [1]. The boundary motion is typically analyzed locally,

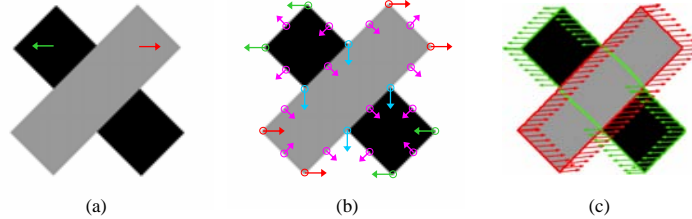

Figure 1: Illustration of the spurious T-junction motion. (a) The front gray bar is moving to the right and the black bar behind is moving to the left [18]. (b) Based on a local window matching, the eight corners of the bars show the correct motion, whereas the T-junctions show spurious downwards motion. (c) Using the boundary-based representation our system is able to correctly estimate the motion and generate the illusory boundary as well.

which can again lead to spurious junction trackings. We are not aware of an existing algorithm that can properly analyze the motions of featureless objects.

In this paper, we use a boundary-based approach which does not rely on motion estimates at corners or junctions. We develop a graphical model which integrates local information and assigns probabilities to candidate contour groupings in order to favor motion interpretations corresponding to the motions of the underlying objects. Boundary completion and discounting the motions of spurious features result from optimizing the graphical model states to explain the contours and their motions. Our system is able to automatically detect and group the boundary fragments, analyze the motion correctly, as well as exploit both static and dynamic cues to synthesize the illusory boundaries (c).

We represent the boundaries at three levels of grouping: *edgelets*, *boundary fragments* and *contours*, where a fragment is a chain of edgelets and a contour is a chain of fragments. Each edgelet within a boundary fragment has a position and an orientation and carries local evidence for motion. The main task of our model is then to group the boundary fragments into contours so that the local motion uncertainties associated with the edgelets are disambiguated and occlusion or other spurious feature events are properly explained. The result is a specialized motion tracking algorithm that properly analyzes the motions of textureless objects.

Our system consists of four conceptual steps, discussed over the next three sections (the last two steps happen together while finding the optimal states in the graphical model):

(a) **Boundary fragment extraction**: Boundary fragments are detected in the first frame.

(b) **Edgelet tracking with uncertainties**: Boundary fragments are broken into edgelets, and, based on local evidence, the probability distribution is found for the motion of each edgelet of each boundary fragment.

(c) **Grouping boundary fragments into contours**: Boundary fragments are grouped, using both temporal and spatial cues.

(d) **Motion estimation**: The final fragment groupings disambiguate motion uncertainties and specify the final inferred motions.

We restrict the problem to two-frame motion analysis though the algorithm can easily be extended to multiple frames.

## 2   Boundary Fragment Extraction

Extracting boundaries from images is a nontrivial task by itself. We use a simple algorithm for boundary extraction, analyzing oriented energy using steerable filters [3] and tracking the boundary in a manner similar to that of the Canny edge detector [2]. A more sophisticated boundary detector can be found in [8]; occluding boundaries can also be detected using special cameras [13]. However, for our motion algorithm designed to handle the special case of textureless objects, we find that our simple boundary detection algorithm works well.

Mathematically, given an image $I$, we seek to obtain a set of fragments $\mathbf{B} = \{\mathbf{b}_i\}$, where each fragment $\mathbf{b}_i$ is a chain of *edgelets* $\mathbf{b}_i = \{e_{ik}\}_{k=1}^{n_i}$. Each edgelet $e_{ik} = \{p_{ik}, \theta_{ik}\}$ is a particle which embeds both location $p_{ik} \in \mathbb{R}^2$ and orientation $\theta_{ik} \in [0, 2\pi)$ information.

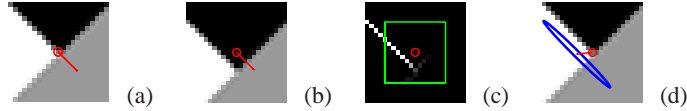

(a)      (b)      (c)      (d)

Figure 2: The local motion vector is estimated for each contour in isolation by selectively comparing orientation energies across frames. (a) A T-junction of the two bar example showing the contour orientation for this motion analysis. (b) The other frame. (c) The relevant orientation energy along the boundary fragment, both for the 2nd frame. A Gaussian pdf is fit to estimate flow, weighted by the oriented energy. (d) Visualization of the Gaussian pdf. The possible contour motions are unaffected by the occluding contour at a different orientation and no spurious motion is detected at this junction.

We use H4 and G4 steerable filters [3] to filter the image and obtain orientation energy per pixel. These filters are selected because they describe the orientation energies well even at corners. For each pixel we find the maximum energy orientation and check if it is local maximum within a slice perpendicular to this orientation. If that is true and the maximum energy is above a threshold $T_1$ we call this point a *primary* boundary point. We collect a pool of primary boundary points after running this test for all the pixels.

We find the primary boundary point with the maximum orientation energy from the pool and do bidirectional contour tracking, consisting of *prediction* and *projection* steps. In the prediction step, the current edgelet generates a new one by following its orientation with a certain step size. In the projection step, the orientation is locally maximized both in the orientation bands and within a small spatial window. The tracking is stopped if the energy is below a threshold $T_2$ or if the turning angle is above a threshold. The primary boundary points that are close to the tracked trajectory are removed from the pool. This process is repeated until the pool is empty. The two thresholds $T_1$ and $T_2$ play the same roles as those in Canny edge detection [2]. While the boundary tracker should stop at sharp corners, it can turn around and continue tracking. We run a postprocess to break the boundaries by detecting points of curvature local maxima which exceed a curvature threshold.

## 3    Edgelet Tracking with Uncertainties

We next break the boundary contours into very short edgelets and obtain the probabilities, based on local motion of the boundary fragment, for the motion vector at each edgelet. We cannot use conventional algorithms, such as Lucas-Kanade [5], for local motion estimation since they rely on corners. The orientation $\theta_{ik}$ for each edgelet was obtained during boundary fragment extraction. We obtain the motion vector by finding the spatial offsets of the edgelet which match the orientation energy along the boundary fragment in this orientation. We fit a Gaussian distribution $\mathcal{N}(\mu_{ik}, \Sigma_{ik})$ of the flow weighted by the orientation energy in the window. The mean and covariance matrix is added to the edgelet: $e_{ik} = \{p_{ik}, \theta_{ik}, \mu_{ik}, \Sigma_{ik}\}$. This procedure is illustrated in Figure 2.

Grouping the boundary fragments allows the motion uncertainties to be resolved. We next discuss the mathematical model of grouping as well as the computational approach.

## 4    Boundary Fragment Grouping and Motion Estimation

### 4.1    Two Equivalent Representations for Fragment Grouping

The essential part of our model is to find the connection between the boundary fragments. There are two possible representations for grouping. One representation is the connection of each end of the boundary fragment. We formulate the probability of this connection to model the *local* saliency of contours. The other equivalent representation is a chain of fragments that forms a contour, on which *global* statistics are formulated, e.g. structural saliency [16]. Similar local and global modeling of contour saliency was proposed in [14]; in [7], both edge saliency and curvilinear continuity were used to extract closed contours from static images. In [15], contour ends are grouped using loopy belief propagation to interpret contours.

The connections between fragment ends are modeled by switch variables. For each boundary fragment $\mathbf{b}_i$, we use a binary variable $\{0, 1\}$ to denote the two ends of the fragment, i.e. $\mathbf{b}_i^{(0)} = e_{i1}$ and $\mathbf{b}_i^{(1)} = e_{i,n_i}$. Let switch variable $S(i, t_i) = (j, t_j)$ denote the connection from $\mathbf{b}_i^{(t_i)}$ to $\mathbf{b}_j^{(t_j)}$. This

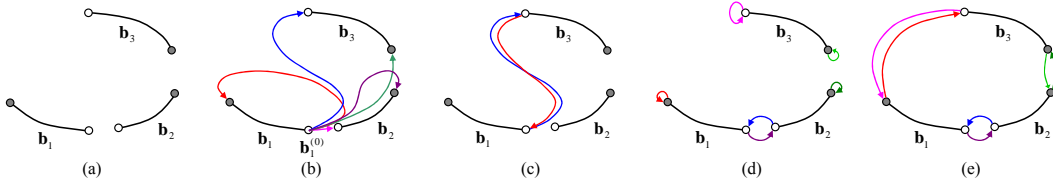

Figure 3: A simple example illustrating switch variables, reversibility and fragment chains. The color arrows show the switch variables. The empty circle indicates end 0 and the filled indicates end 1. (a) Shows three boundary fragments. Theoretically $\mathbf{b}_1^{(0)}$ can connect to any of the other ends including itself, (b). However, the switch variable is *exclusive*, i.e. there is only one connection to $\mathbf{b}_1^{(0)}$, and *reversible*, i.e. if $\mathbf{b}_1^{(0)}$ connects to $\mathbf{b}_3^{(0)}$, then $\mathbf{b}_3^{(0)}$ should also connect to $\mathbf{b}_1^{(0)}$, as shown in (c). Figures (d) and (e) show two of the legal contour groupings for the boundary fragments: two open contours and a closed loop contour.

connection is *exclusive*, i.e. each end of the fragment should either connect to one end of the other fragment, or simply have no connection. An exclusive switch is further called *reversible*, i.e.

$$\text{if } S(i, t_i) = (j, t_j), \text{ then } S(j, t_j) = (i, t_i),$$

or in a more compact form

$$S(S(i, t_i)) = (i, t_i). \tag{1}$$

When there is no connection to $\mathbf{b}_i^{(t_i)}$, we simply set $S(i, t_i) = (i, t_i)$. We use the binary function $\delta[S(i, t_i) - (j, t_j)]$ to indicate whether there is a connection between $\mathbf{b}_i^{(t_i)}$ and $\mathbf{b}_j^{(t_j)}$. The set of all the switches are denoted as $\mathbf{S} = \{S(i, t_i) | i = 1 : N, t_i = 0, 1\}$. We say $\mathbf{S}$ is reversible if every switch variable satisfies Eqn. (1). The reversibility of switch variables is shown in Figure 3 (b) and (c).

From the values of the switch variables we can obtain contours, which are chains of boundary fragments. A fragment chain is defined as a series of the end points $\mathbf{c} = \{(\mathbf{b}_{i_1}^{(x_1)}, \mathbf{b}_{i_1}^{(\overline{x}_1)}), \cdots, (\mathbf{b}_{i_m}^{(x_m)}, \mathbf{b}_{i_m}^{(\overline{x}_m)})\}$. The chain is specified by fragment label $\{i_1, \cdots, i_m\}$ and end label $\{x_1, \cdots, x_m\}$. It can be either open or closed. The order of the chain is determined by the switch variable. Each end appears in the chain at most once. The notation of a chain is not unique. Two open chains are identical if the fragment and end labels are reversed. Two closed chains are identical if they match each other by rotating one of them. These identities are guaranteed from the reversibility of the switch variables. A set of chains $\mathbf{C} = \{c_i\}$ can be uniquely extracted based on the values of the reversible switch variables, as illustrated in Figure 3 (d) and (e).

## 4.2 The Graphical Model

Given the observation $O$, the two images, and the boundary fragments $\mathbf{B}$, we want to estimate the flow vectors $\mathbf{V} = \{\mathbf{v}_i\}$ and $\mathbf{v}_i = \{v_{ik}\}$, where each $v_{ik}$ associates with edgelet $e_{ik}$, and the grouping variables $\mathbf{S}$ (switches) or equivalently $\mathbf{C}$ (fragment chains). Since the grouping variable $\mathbf{S}$ plays an essential role in the problem, we shall first infer $\mathbf{S}$ and then infer $\mathbf{V}$ based on $\mathbf{S}$.

### 4.2.1 The Graph for Boundary Fragment Grouping

We use two equivalent representations for boundary grouping, switch variables and chains. We use $\delta[S(S(i, t_i)) - (i, t_i)]$ for each end to enforce the reversibility. Suppose otherwise $S(i_1, t_{i_1}) = S(i_2, t_{i_2}) = (j, t_j)$ for $i_1 \neq i_2$. Let $S(j, t_j) = (i_1, t_{i_1})$ without loss of generality, then $\delta[S(S(i_2, t_{i_2})) - (i_2, t_{i_2})] = 0$, which means that the switch variables are not reversible.

We use a function $\lambda(S(i, t_i); \mathbf{B}, O)$ to measure the distribution of $S(i, t_i)$, i.e. how likely $\mathbf{b}_i^{(t_i)}$ connects to the end of other fragments. Intuitively, two ends should be connected if

◇ **Motion similarity** the distributions of the motion of the two end edgelets are similar;

◇ **Curve smoothness** the illusory boundary to connect the two ends is smooth;

◇ **Contrast consistency** the brightness contrast at the two ends consistent with each other.

We write $\lambda(\cdot)$ as a product of three terms, one enforcing each criterion. We shall follow the example in Figure 4 to simplify the notation, where the task is to compute $\lambda(S(1, 0) = (2, 0))$. The first term

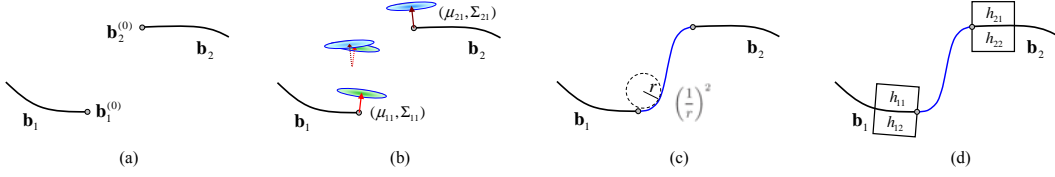

(a)           (b)           (c)           (d)

Figure 4: An illustration of local saliency computation. (a) Without loss of generalization we assume the two ends to be $b_1^{(0)}$ and $b_2^{(0)}$. (b) The KL divergence between the distributions of flow vectors are used to measure the motion similarity. (c) An illusory boundary $\gamma$ is generated by minimizing the energy of the curve. The sum of square curvatures are used to measure the curve smoothness. (d) The means of the local patches located at the two ends are extracted, i.e. $h_{11}$ and $h_{12}$ from $b_1^{(0)}$, $h_{21}$ and $h_{22}$ from $b_2^{(0)}$, to compute contrast consistency.

is the KL divergence between the two Gaussian distributions of the flow vectors

$$\exp\{-\alpha_{KL}KL(\mathcal{N}(\mu_{11}, \Sigma_{11}), \mathcal{N}(\mu_{21}, \Sigma_{21}))\}, \tag{2}$$

where $\alpha_{KL}$ is a scaling factor. The second term is the local saliency measure on the illusory boundary $\gamma$ that connects the two ends. The illusory boundary is simply generated by minimizing the energy of the curve. The saliency is defined as

$$\exp\left\{-\alpha_\gamma \int_\gamma \left(\frac{d\theta}{ds}\right)^2 ds\right\}, \tag{3}$$

where $\theta(s)$ is the slope along the curve, and $\frac{d\theta}{ds}$ is local curvature [16]. $\alpha_\gamma$ is a scaling factor. The third term is computed by extracting the mean of local patches located at the two ends

$$\exp\left\{-\frac{d_{max}}{2\sigma_{max}^2} - \frac{d_{min}}{2\sigma_{min}^2}\right\}, \tag{4}$$

where $d_1 = (h_{11} - h_{21})^2$, $d_2 = (h_{12} - h_{22})^2$, and $d_{max} = \max(d_1, d_2)$, $d_{min} = \min(d_1, d_2)$. $\sigma_{max} > \sigma_{min}$ are the scale parameters. $h_{11}$, $h_{12}$, $h_{21}$, $h_{22}$ are the means of the pixel values of the four patches located at the two end points. For self connection we simply set a constant value: $\lambda(S(i, t_i) = (i, t_i)) = \tau$.

We use a function $\psi(\mathbf{c}_i; \mathbf{B}, O)$ to model the structural saliency of contours. It was discovered in [10] that convex occluding contours are more salient, and additional T-junctions along the contour may increase or decrease the occlusion perception. Here we simply enforce that a contour should have no self-intersection. $\psi(\mathbf{c}_i; \mathbf{B}, O) = 1$ if there is no self intersection and $\psi(\mathbf{c}_i; \mathbf{B}, O) = 0$ otherwise.

Thus, the (discrete) graphical model favoring the desired fragment grouping is

$$\Pr(\mathbf{S}; B, O) = \frac{1}{Z_S} \prod_{i=1}^N \prod_{t_i=0}^1 \lambda(S(i, t_i); \mathbf{B}, O)\delta[S(S(i, t_i)) - (i, t_i)] \cdot \prod_{j=1}^M \psi(\mathbf{c}_j; \mathbf{B}, O), \tag{5}$$

where $Z_S$ is a normalization constant. Note that this model measures both the switch variables $S(i, t_i)$ for local saliency and the fragment chains $\mathbf{c}_i$ to enforce global structural saliency.

### 4.2.2 Gaussian MRF on Flow Vectors

Given the fragment grouping, we model the flow vectors $\mathbf{V}$ as a Gaussian Markov random field (GMRF). The edgelet displacement within each boundary fragment should be smooth and match the observation along the fragment. The probability density is formulated as

$$\varphi(\mathbf{v}_i; \mathbf{b}_i) = \prod_{k=1}^{n_i} \exp\{-(v_{ik} - \mu_{ik})^T \Sigma_{ik}^{-1}(v_{ik} - \mu_{ik})\} \prod_{k=1}^{n_i-1} \exp\{-\frac{1}{2\sigma^2}\|v_{ik} - v_{i,k+1}\|^2\}, \tag{6}$$

where $\mu_{ik}$ and $\Sigma_{ik}$ are the motion parameters of each edgelet estimated in Sect 3.

We use $\mathbf{V}(i, t_i)$ to denote the flow vector of end $t_i$ of fragment $\mathbf{b}_i$. We define $\mathbf{V}(S(i, t_i)) = \mathbf{V}(j, t_j)$ if $S(i, t_i) = (j, t_j)$. Intuitively the flow vectors of the two ends should be similar if they are connected, or mathematically

$$\phi(\mathbf{V}(i, t_i), \mathbf{V}(S(i, t_i))) = \begin{cases} 1 & \text{if } S(i, t_i) = (i, t_i), \\ \exp\{-\frac{1}{2\sigma^2}\|\mathbf{V}(i, t_i) - \mathbf{V}(S(i, t_i))\|^2\} & \text{otherwise.} \end{cases} \tag{7}$$

The (continuous) graphical model of the flow vectors is therefore defined as

$$\Pr(\mathbf{V}|\mathbf{S};\mathbf{B}) = \frac{1}{Z_V} \prod_{i=1}^{N} \varphi(\mathbf{v}_i;\mathbf{b}_i) \prod_{t_i=0}^{1} \phi(\mathbf{V}(i,t_i), \mathbf{V}(S(i,t_i))) \qquad (8)$$

where $Z_V$ is a normalization constant. When $\mathbf{S}$ is given it is a GMRF which can be solved by least squares.

## 4.3 Inference

Having defined the graphical model to favor the desired motion and grouping interpretations, we need to find the state parameters that best explain the image observations. The natural decomposition of $\mathbf{S}$ and $\mathbf{V}$ in our graphical model

$$\Pr(\mathbf{V}, \mathbf{S};\mathbf{B}, O) = \Pr(\mathbf{S};\mathbf{B}, O) \cdot \Pr(\mathbf{V}|\mathbf{S};\mathbf{B}, O), \qquad (9)$$

(where $\Pr(\mathbf{S};\mathbf{B}, O)$ and $\Pr(\mathbf{V}|\mathbf{S};\mathbf{B}, O)$ are defined in Eqn. (5) and (8) respectively) lends itself to performing two-step inference. We first infer the boundary grouping $\mathbf{B}$, and then infer $\mathbf{V}$ based on $\mathbf{B}$. The second step is simply to solve least square problem since $\Pr(\mathbf{V}|\mathbf{S};\mathbf{B}, O)$ is a GMRF. This approach does not globally optimize Eqn. (9) but results in reasonable solution because $\mathbf{V}$ strongly depends on $\mathbf{S}$. The density function $\Pr(\mathbf{S};\mathbf{B}, O)$ is not a random field, so we use importance sampling [6] to obtain the marginal distribution $\Pr(S(i, t_i);\mathbf{B}, O)$. The proposal density of each switch variable is set to be

$$q\left(S(i, t_i) = (j, t_j)\right) \propto \frac{1}{Z_q}\lambda\left(S(i, t_i) = (j, t_j)\right)\lambda\left(S(j, t_j) = (i, t_i)\right) \qquad (10)$$

where $\lambda(\cdot)$ has been normalized to sum to 1 for each end. We found that this bidirectional measure is crucial to take valid samples. To sample the proposal density, we first randomly select a boundary fragment, and connect to other fragments based on $q(S(i, t_i))$ to form a contour (a chain of boundary fragments). Each end is sampled only once, to ensure reversibility. This procedure is repeated until no fragment is left. In the importance step we run the binary function $\psi(\mathbf{c}_i)$ to check that each contour has no self-intersection. If $\psi(\mathbf{c}_i) = 0$ then this sample is rejected. The marginal distributions are estimated from the samples. Lastly the optimal grouping is obtained by replacing random sampling with selecting the maximum-probability connection over the estimated marginal distributions. The number of samples needed depends on the number of the fragments. In practice we find that $n^2$ samples are sufficient for $n$ fragments.

## 5  Experimental Results

Figure 6 shows the boundary extraction, grouping, and motion estimation results of our system for both real and synthetic examples[1]. All the results are generated using the same parameter settings. The algorithm is implemented in MATLAB, and the running time varies from ten seconds to a few minutes, depending on the number of the boundary fragments found in the image.

The two-bar examples in Figure 1(a) yields fourteen detected boundary fragments in Figure 6(a) and two contours in (b). The estimated motion matches the ground truth at the T-junctions. The fragments belonging to the same contour are plotted in the same color and the illusory boundaries are synthesized as shown in (c). The boundaries are warped according to the estimated flow and displayed in (d). The hallucinated illusory boundaries in frame 1 (c) and 2 (d) are plausible amodal completions.

The second example is the Kanizsa square where the frontal white square moves to the right bottom. Twelve fragments are detected in (a) and five contours are grouped in (b). The estimated motion and generated illusory boundary also match the ground truth and human perception. Notice that the arcs tend to connect to other ones if we do not impose the structural saliency $\psi(\cdot)$.

We apply our system to a video of a dancer (Figure 5 (a) and (b)). In this stimulus the right leg moves downwards, but there is weak occluding boundary at the intersection of the legs. Eleven

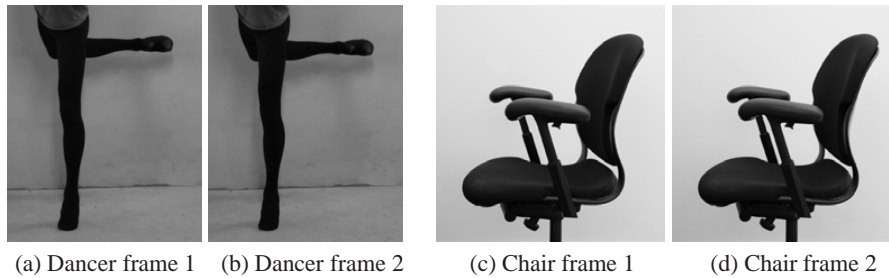

(a) Dancer frame 1    (b) Dancer frame 2      (c) Chair frame 1       (d) Chair frame 2

Figure 5: Input images for the non-synthetic examples of Figure 6. The dancer's right leg is moving downwards and the chair is rotating (note the changing space between the chair's arms).

boundary fragments are extracted in (a) and five contours are extracted in (b). The estimated motion (b) matches the ground truth. The hallucinated illusory boundary in (c) and (d) correctly connect the occluded boundary of the right leg and the invisible boundary of the left leg.

The final row shows challenging images of a rotating chair (Figure 5 (c) and (d)), also showing proper contour completion and motion analysis. Thirty-seven boundary fragments are extracted and seven contours are grouped. To complete the occluded contours of this image would be nearly impossible working only from a static image. Exploiting motion as well as static information, our system is able to complete the contours properly.

Note that the traditional motion analysis algorithms fail at estimating motion for these examples (see supplementary videos) and would thus also fail at correctly grouping the objects based on the motion cues.

## 6   Conclusion

We propose a novel boundary-based representation to estimate motion under the challenging visual conditions of moving textureless objects. Ambiguous local motion measurements are resolved through a graphical model relating edgelets, boundary fragments, completed contours, and their motions. Contours are grouped and their motions analyzed simultaneously, leading to the correct handling of otherwise spurious occlusion and T-junction features. The motion cues help the contour completion task, allowing completion of contours that would be difficult or impossible using only low-level information in a static image. A motion analysis algorithm such as this one that correctly handles featureless contour motions is an essential element in a visual system's toolbox of motion analysis methods.

## Footnotes

[1]The results can be viewed online http://people.csail.mit.edu/celiu/contourmotions/

## References

[1] M. J. Black and D. J. Fleet. Probabilistic detection and tracking of motion boundaries. *International Journal of Computer Vision*, 38(3):231–245, 2000.

[2] J. Canny. A computational approach to edge detection. *IEEE Trans. Pat. Anal. Mach. Intel.*, 8(6):679–698, Nov 1986.

[3] W. T. Freeman and E. H. Adelson. The design and use of steerable filters. *IEEE Trans. Pat. Anal. Mach. Intel.*, 13(9):891–906, Sep 1991.

[4] B. K. P. Horn and B. G. Schunck. Determing optical flow. *Artificial Intelligence*, 17:185–203, 1981.

[5] B. Lucas and T. Kanade. An iterative image registration technique with an application to stereo vision. In *Proceedings of the International Joint Conference on Artificial Intelligence*, pages 674–679, 1981.

[6] D. Mackay. *Information Theory, Inference, and Learning Algorithms*. Cambridge University Press, 2003.

[7] S. Mahamud, L. Williams, K. Thornber, and K. Xu. Segmentation of multiple salient closed contours from real images. *IEEE Trans. Pat. Anal. Mach. Intel.*, 25(4):433–444, 2003.

[8] D. Martin, C. Fowlkes, and J. Malik. Learning to detect natural image boundaries using local brightness, color, and texture cues. *IEEE Trans. Pat. Anal. Mach. Intel.*, 26(5):530–549, May 2004.

[9] J. McDermott. Psychophysics with junctions in real images. *Perception*, 33:1101–1127, 2004.

[10] J. McDermott and E. H. Adelson. The geometry of the occluding contour and its effect on motion interpretation. *Journal of Vision*, 4(10):944–954, 2004.

[11] J. McDermott and E. H. Adelson. Junctions and cost functions in motion interpretation. *Journal of Vision*, 4(7):552–563, 2004.

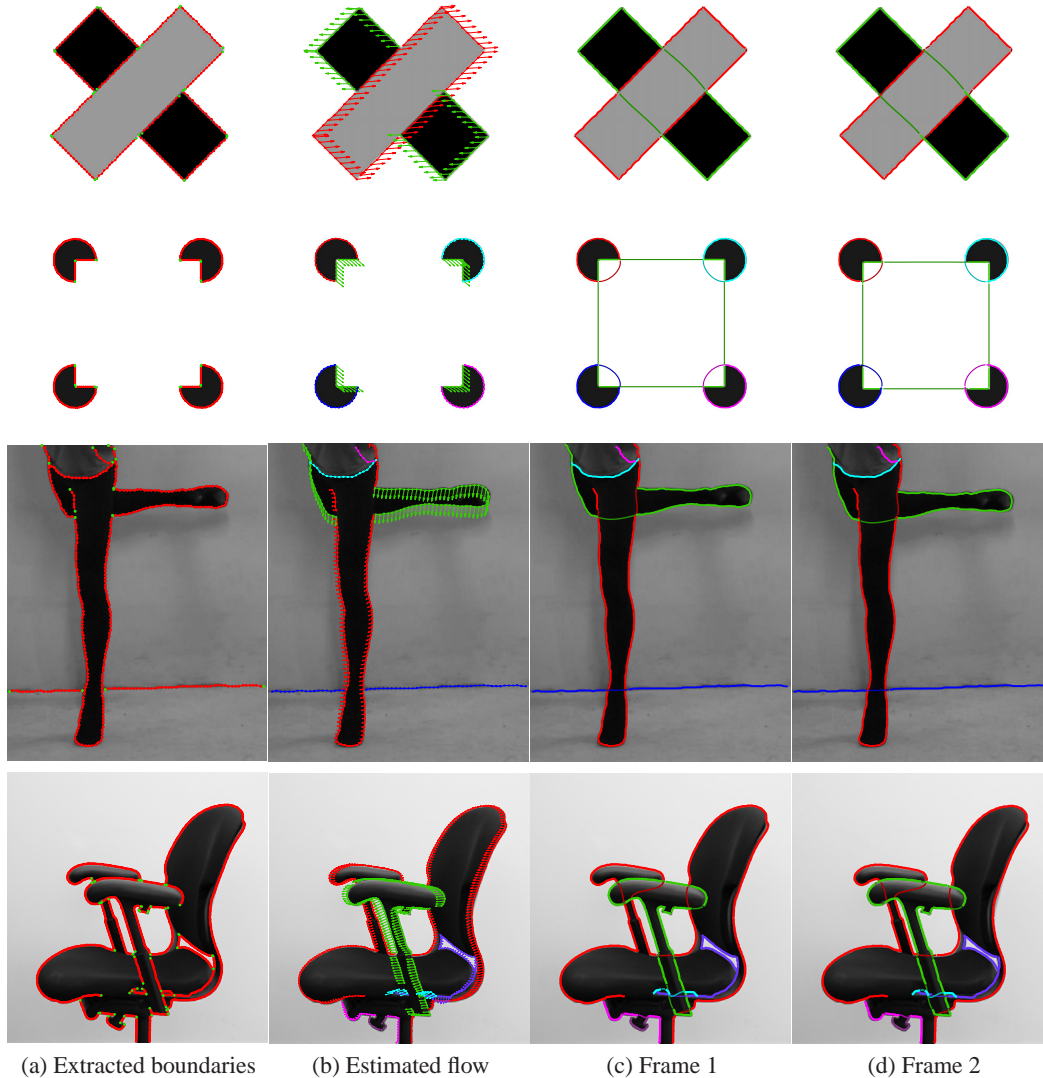

| (a) Extracted boundaries | (b) Estimated flow | (c) Frame 1 | (d) Frame 2 |

Figure 6: Experimental results for some synthetic and real examples. The same parameter settings were used for all examples. Column (a): Boundary fragments are extracted using our boundary tracker. The red dots are the edgelets and the green ones are the boundary fragment ends. Column (b): Boundary fragments are grouped into contours and the flow vectors are estimated. Each contour is shown in its own color. Columns (c): the illusory boundaries are generated for the first and second frames. The gap between the fragments belonging to the same contour are linked exploiting both static and motion cues in Eq. (5).

[12] S. J. Nowlan and T. J. Sejnowski. A selection model for motion processing in area mt primates. *The Journal of Neuroscience*, 15(2):1195–1214, 1995.

[13] R. Raskar, K.-H. Tan, R. Feris, J. Yu, and M. Turk. Non-photorealistic camera: depth edge detection and stylized rendering using multi-flash imaging. *ACM Trans. Graph. (SIGGRAPH)*, 23(3):679–688, 2004.

[14] X. Ren, C. Fowlkes, and J. Malik. Scale-invariant contour completion using conditional random fields. In *Proceedings of International Conference on Computer Vision*, pages 1214–1221, 2005.

[15] E. Saund. Logic and MRF circuitry for labeling occluding and thinline visual contours. In *Advances in Neural Information Processing Systems 18*, pages 1153–1160, 2006.

[16] A. Shahua and S. Ullman. Structural saliency: the detection of globally salient structures using a locally connected network. In *Proceedings of International Conference on Computer Vision*, pages 321–327, 1988.

[17] J. Shi and C. Tomasi. Good features to track. In *IEEE Conference on Computer Vision and Pattern Recognition*, pages 593–600, 1994.

[18] Y. Weiss and E. H. Adelson. Perceptually organized EM: A framework for motion segmentaiton that combines information about form and motion. Technical Report 315, M.I.T Media Lab, 1995.
